# The Tempo 2 Algorithm: Adjusting Time-Delays By Supervised Learning

**Ulrich Bodenhausen and Alex Waibel**
School of Computer Science
Carnegie Mellon University
Pittsburgh, PA 15213

## Abstract

In this work we describe a new method that adjusts time-delays and the widths of time-windows in artificial neural networks automatically. The input of the units are weighted by a gaussian input-window over time which allows the learning rules for the delays and widths to be derived in the same way as it is used for the weights. Our results on a phoneme classification task compare well with results obtained with the TDNN by Waibel et al., which was manually optimized for the same task.

## 1 INTRODUCTION

The processing of pattern-sequences has been investigated with several neural network architectures. One approach to processing of temporal context with neural networks is to implement time-delays. This approach is neurophysiologically plausible, because real axons have a limited conduction speed (which is dependent on the diameter of the axon and whether it is myelinated or not). Additionally, the length of most axons is much greater than the euclidean distance between the connected neurons. This leads to a great variety of different time-delays in the brain. Artificial networks that make use of time-delays have been suggested [10, 11, 12, 8, 2, 3].

In the TDNN [11, 12] and most other artificial neural networks with time-delays the delays are implemented as hat-shaped input-windows over time. A unit j that is connected with unit i by a connection with delay n is only receiving information about the activity of unit i n time-steps ago. A set of connections with consecutive time-delays is used to let each unit gather a certain amount of temporal context. In these networks, weights are automatically trained but the architecture of the network (time-delays, number of connections and number of units) have to be predetermined by laborious experiments [8, 6].

In this work we describe a new algorithm that adjusts time-delays and the width of the input-window automatically. The learning rules require input-windows over time that can be described by a smooth function. With these input-windows it is possible to derive learning rules for adjusting the center and the width of the window. During training, new connections are added if they are needed by splitting already existing connections and training them independently.

Adaptive time-delays in neural networks could have significant advantages for the processing of pattern-sequences, especially if the relevant information is distributed across non-consecutive patterns. A typical example for this kind of pattern sequences are rhythms (relevant in music and speech). In a rhythm, there are many events but also many gaps between these events. Another example is speech, where some parts of an utterance are more important for understanding than others (example: 'hat', 'fat', 'cat'..). Therefore a network that allocates existing and new resources to the parts of the input sequence that are most helpful for the task could be more compact and efficient for various tasks.

## 2   THE TEMPO 2 NETWORK

The Tempo 2 network is an artificial neural network with *adaptive* weights, *adaptive* time-delays and *adaptive* widths of gaussian input windows over time. It is a generalization of the Back-Propagation network proposed by Rumelhart, Hinton and Williams [9]. The network is based on some ideas that were tested with the Tempo 1 network [2, 3].

The Tempo 2 network is designed to learn about the relevant temporal context during training. A unit in the network is activated by input from a gaussian shaped input-window centered around (t-d) and standard deviation $\sigma$, where d (the time-delay) and $\sigma$ (the width of the input-window) are to be learned [1] (see Fig. 1 and 2). This means that the center and the width of each input-window can be adjusted by learning rules. The adaptive time-delays allow the processing of temporal context that is distributed across several non-consecutive patterns of the sequence. The adaptive width of the window enables the receiving unit to monitor a variable sequence of consecutive activations over time of each sending unit. New connections can be added if they are needed (see section 2.1). The input of unit j at time t, $x(t)_j$, is

$$x(t)_j = \sum_{\tau=0}^{t} \sum_{k} y_k(\tau)\theta(\tau, t, d_{jk}, \sigma_{jk})w_{jk}$$

with $\theta(\tau, t, d_{jk}, \sigma_{jk})$ representing the gaussian input window given by

$$\theta(\tau, t, d_{jk}, \sigma_{jk}) = \frac{1}{\sqrt{2\pi}\sigma_{jk}}e^{(\tau-t+d_{jk})^2/2\sigma_{jk}^2}$$

where $y_k$ is the output of the previous sending unit and $w_{jk}$, $d_{jk}$ and $\sigma_{jk}$ are the weights, delays and widths on its connections, respectively.

This approach is partly motivated by neurophysiology and mathematics. In the brain, a spike that is sent by a neuron via an axon is not received as a spike by the receiving cell.

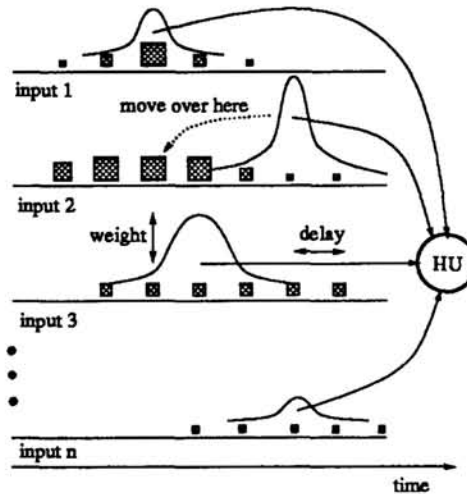

Figure 1: The input to one unit in the Tempo 2 network. The boxes represent the activations of the sending units; a tall box represents a high activity.

Rather, the postsynaptic potential has a short rise and a long tail. Let us assume a situation with two neurons. Neuron A fires at time t-d, where d is the time that the signal needs to travel along the connection and to activate neuron B. Neuron B is activated mostly at time t, but the postsynaptic potential will decrease slowly and neuron B will get some input at time t+1, some smaller input at time t+2 and so on. Functionally, a spike is smeared over time and this provides some "local memory".

For our simulations we simulate this behavior by allowing the receiving unit to be activated by the weighted sum of activations around an input centered at time t-d. If the sending unit ("neuron A") was activated at time t-d, then the receiving unit ("neuron B") will be activated mostly at time t, will be less activated at time t+1, and so on. In our case, the input-window function also allows the receiving unit to be (less) activated at times t-1, t-2 etc.. This enables us to formulate a learning rule that can increase and decrease time-delays.

The gaussian input-window has the advantage that it provides some robustness against temporally misaligned input tokens. By looking at Fig. 2 it is obvious that small misalignments of the input signal do not change the input of the receiving unit significantly. The robustness is dependent on the width of the window. Therefore a wide window would make the input of the receiving unit more robust against signals shifted in time, but would also reduce the time-resolution of the unit. This suggests the implementation of a learning rule that adjusts the width of the input-windows of each connection.

With this gaussian input-window, it is possible to compute how the input of unit j would change if the delay of a connection or the width of the input window were changed. The formalism is the same as for the derivation of the learning rules for the weights in a standard Back-Propagation network. The change of a delay is proportional to the derivative of the output error with respect to the delay. The change of a width is proportional to the derivative of the error with respect to the width. The error at the output is propagated back to the hidden layer. The learning rules for weights $w_{ji}$, delays $d_{ji}$ and widths $\sigma_{ji}$ were derived from

$$\Delta w_{ji} = -\epsilon_1 \frac{\partial E}{\partial w_{ji}}$$

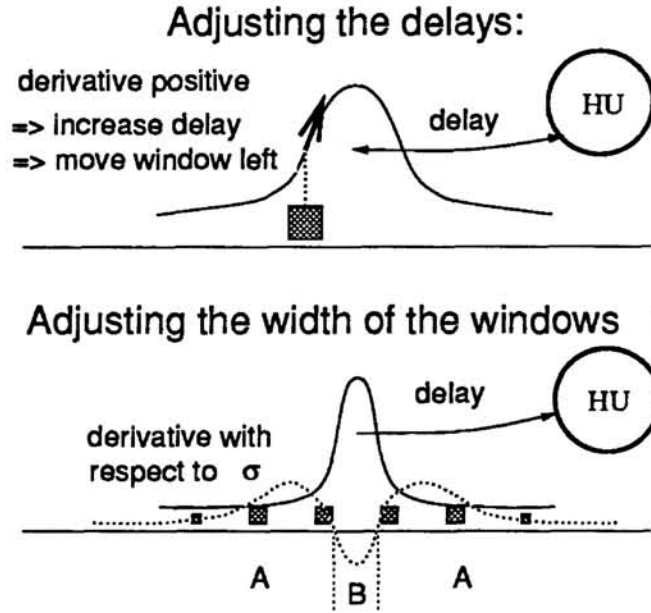

Figure 2: A graphical explanation of the learning rules for delays and widths: The derivative of the gaussian input-window with respect to time is used for adjusting the time-delays. The derivative with respect to $\sigma$ (dotted line) is used for adjusting the width of the window. A majority of activation in area A will cause the window to grow. A majority of activity in area B will cause the window to shrink.

$$\Delta d_{ji} = -\epsilon_2 \frac{\partial E}{\partial d_{ji}}$$

$$\Delta \sigma_{ji} = -\epsilon_3 \frac{\partial E}{\partial \sigma_{ji}}$$

where $\epsilon_1$, $\epsilon_2$ and $\epsilon_3$ are the learning rates and E is the error. As in the derivation of the standard Back-Propagation learning rules, the chain rule is applied ($z = w, d, \sigma$):

$$\frac{\partial E}{\partial z_{ji}} = \frac{\partial E}{\partial x(t)_j} \frac{\partial x(t)_j}{\partial z_{ji}}$$

where $\frac{\partial E}{\partial x(t)_j}$ is the same in the learning rules for weights, delays and widths. The partial derivatives of the input with respect to the parameters of the connections are computed as follows:

$$\frac{\partial x(t)_j}{\partial w_{ji}} = \sum_{\tau=0}^{t} y_i(\tau)\theta(\tau, t, d_{ji}, \sigma_{ji})$$

$$\frac{\partial x(t)_j}{\partial d_{ji}} = \sum_{\tau=0}^{t} y_i(\tau)w_{ji}\frac{\partial}{\partial d_{ji}}\theta(\tau, t, d_{ji}, \sigma_{ji})$$

## Splitting the Connections:

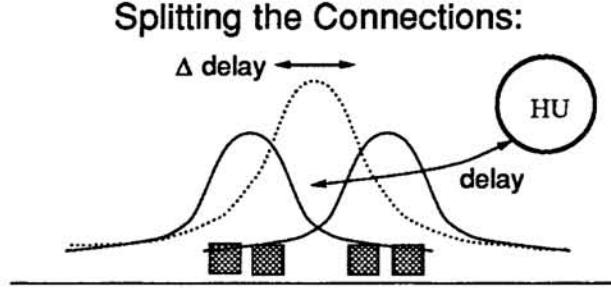

Figure 3: Splitting of a connection. The dotted line represents the "old" window and the solid lines represent the two windows after splitting, respectively.

$$\frac{\partial x(t)_j}{\partial \sigma_{ji}} = \sum_{\tau=0}^{t} y_i(\tau) w_{ji} \frac{\partial}{\partial \sigma_{ji}} \theta(\tau, t, d_{ji}, \sigma_{ji})$$

### 2.1 ADDING NEW CONNECTIONS

Learning algorithms for neural networks that add hidden units have recently been proposed [4, 5]. In our network connections are added to the already existing ones in a similar way as it is used by Hanson for adding units [5]. During learning, the network starts with one connection between two units. Depending on the task this may be insufficient and it would be desirable to add new connections where more connections are *needed*. New connections are added by splitting already existing connections and afterwards training them independently (see Fig. 3). The rule for splitting a connection is motivated by observations during training runs. It was observed that input-windows started moving backwards and forwards (that means the time-delays changed) after a certain level of performance was reached. This can be interpreted as inconsistent time-delays which might be caused by temporal variability of certain features in the samples of speech. During training we compute the standard deviations of all delay changes and compare them with a threshold:

$$if \sum_{alltokens} (\Delta d_{ji}(token) - \frac{\sum_{alltokens} |\Delta d_{ji}|}{\#tokens})^2 > threshold$$

then split connection ji.

## 3  SIMULATIONS

The Tempo 2 network was initially tested with rhythm classification. The results were encouraging and evaluation was carried out on a phoneme classification task. In this application, adaptive delays can help to find important cues in a sample of speech. Units should not accumulate information from irrelevant parts of the phonemes. Rather, they should look at parts within the phonemes that provide the most important information for the kind of feature extraction that is needed for the classification task. The network was trained on the phonemes /b/, /d/ and /g/ from a single speaker. 783 tokens were used for training and 759 tokens were used for testing.

| adaptive parameters | constant parameters | Training Set | Testing Set |
|---|---|---|---|
| weights | delays, widths | 93.2% | 89.3% |
| delays | weights, widths | 64.0% | 63.0% |
| widths | weights, delays | 63.5% | 61.8% |
| delays, widths | weights | 70.0% | 68.6% |
| weights, delays | widths | 98.3% | 97.8% |
| weights, delays, widths | - | 98.8% | 98.0% |

Table 1: /b/, /d/ and /g/ classification performance with 8 hidden units in one hidden layer. The network is initialized with random weights and constant widths.

In order to evaluate the usefulness of each adaptive parameter, the network was trained and tested with a variety of combinations of constant and adaptive parameters (see Table 1). In all cases the network was initialized with random weights and delays and constant widths $\sigma$ of the input windows. All results were obtained with 8 hidden units in one hidden layer.

## 4   DISCUSSION

The TDNN has been shown to be a very powerful approach to phoneme recognition. The fixed time-delays and the kind of time-window were chosen partly because they were motivated by results from earlier studies [1, 7] and because they were successful from an engineering point of view. The architecture was optimized for the recognition of phonemes /b/, /d/ and /g/ and could be applied to other phonemes without significant changes. In this study we explored the performance of an artificial neural network that can automatically *learn* its own architecture by learning time-delays and widths of the gaussian input windows. The learning rules for the time-delays and the width of the windows were derived in the same way that has been shown successful for the derivation of learning rules for weights.

Our results show that time-delays in artificial neural networks can be learned automatically. The learning rule proposed in this study is able to improve performance significantly compared to fixed delays if the network is initialized with random delays.

The width of an input window determines how much local temporal context is captured by a single connection. Additionally, a large window means increased robustness against temporal misalignments of the input tokens. A large window also means that the connection transmits with a low temporal resolution. The learning rule for the widths of the windows has to compromise between increased robustness against misaligned tokens and decreased time-resolution. This is done by a gradient descent method.

If the network is initialized with the same widths that are used for the training runs with constant widths, 70 - 80% of the windows in the network get smaller during training. Our simulations show that it is possible to let a learning rule adjust parameters that determine the temporal resolution of the network.

The comparison of the performances with one adaptive parameter set (either weights, delays or widths) shows that the main parameters in the network are the weights. Delays and widths seem to be of a lesser importance, but in combination with the weights the delays can improve the performance, especially generalization. A Tempo 2 network with trained delays and widths and *random* weights can classify 70% of the phonemes correctly.

This suggests that learning temporal parameters is effective.

The network achieves results comparable to a similar network with a handtuned architecture. This suggests that the kind of learning rule could be helpful in applying time-delay neural networks to problems where no knowledge about optimal time windows is available. At higher levels of processing such adaptive networks could be used to learn rhythmic (prosodic) relationships in fluent speech and other tasks.

### Acknowledgements

The authors gratefully acknowledge the support by the McDonnel-Pew Foundation (Cognitive Neuroscience Program) and ATR Interpreting Telephony Research Laboratories.

## Footnotes

[1]Other windows are possible. The function describing the shape of the window has to be smooth.

## References

[1] S.E. Blumstein and K.N. Stevens. Perceptual Invariance And Onset Spectra For Stop Consonants In Different Vowel Environments. *Journal of the Acoustical Society of America*, 67:648–662, 1980.

[2] U. Bodenhausen. The Tempo Algorithm: Learning In A Neural Network With Adaptive Time-Delays. In *Proceedings of the IJCNN 90, Washington D.C.*, January 1990.

[3] U. Bodenhausen. Learning Internal Representations Of pattern Sequences In A Neural Network With Adaptive Time-Delays. In *Proceedings of the IJCNN 90, San Diego*, June 1990.

[4] S. Fahlman and C. Lebiere. The Cascade-Correlation Learning Architecture. In *Advances in Neural Information Processing Systems*. Morgan Kaufmann, 1990.

[5] S. J. Hanson. Meiosis Networks. In *Advances in Neural Information Processing Systems*. Morgan Kaufmann, 1990.

[6] Kamm, C. E.. Effects Of Neural Network Input Span On Phoneme Classification. In *Proceedings of the International Joint Conference on Neural Networks*, June 1990.

[7] D. Kewley-Port. Time Varying Features As Correlates Of Place Of Articulation In Stop Consonants. *Journal of the Acoustical Society of America*, 73:322–335, 1983.

[8] K. J. Lang, G. E. Hinton, and A.H. Waibel. A Time-Delay Neural Network Architecture For Speech Recognition. *Neural Networks Journal*, 1990.

[9] D. E. Rumelhart, G. E. Hinton, and R.J. Williams. Learning Internal Representations By Error Propagation. In J.L. McClelland and D.E. Rumelhart, editors, *Parallel Distributed Processing; Explorations in the Microstructure of Cognition*, chapter 8, pages 318–362. MIT Press, Cambridge, MA, 1986.

[10] D.W. Tank and J.J. Hopfield. Neural Computation By Concentrating Information In Time. In *Proceedings National Academy of Sciences*, pages 1896–1900, April 1987.

[11] A. Waibel, T. Hanazawa, G. Hinton, K. Shikano, and K. Lang. Phoneme Recognition Using Time-Delay Neural Networks. *IEEE, Transactions on Acoustics, Speech and Signal Processing*, March 1989.

[12] A. Waibel. Modular Construction Of Time-Delay Neural Networks For Speech Recognition. *Neural Computation, MIT-Press*, March 1989.
